# Multi Dimensional ICA to Separate Correlated Sources

**Roland Vollgraf, Klaus Obermayer**
Department of Electrical Engineering and Computer Science
Technical University of Berlin Germany
{*vro,oby*} *@cs.tu-berlin.de*

## Abstract

We present a new method for the blind separation of sources, which do not fulfill the independence assumption. In contrast to standard methods we consider groups of neighboring samples ("patches") within the observed mixtures.

First we extract independent features from the observed patches. It turns out that the average dependencies between these features in different sources is in general lower than the dependencies between the amplitudes of different sources. We show that it might be the case that most of the dependencies is carried by only a small number of features. Is this case - provided these features can be identified by some heuristic - we project all patches into the subspace which is orthogonal to the subspace spanned by the "correlated" features.

Standard ICA is then performed on the elements of the transformed patches (for which the independence assumption holds) and robustly yields a good estimate of the mixing matrix.

## 1 Introduction

ICA as a method for blind source separation has been proven very useful in a wide range of statistical data analysis. A strong criterion, that allows to detect and separate linearly mixed source signals from the observed mixtures, is the independence of the source signals amplitude distribution. Many contrast functions rely on this assumption, e.g. in the way, that they estimate the Kullback-Leibler distance to a (non-Gaussian) factorizing multivariate distribution [1, 2, 3]. Others consider higher order moments of the source estimates [4, 5]. Naturally these algorithms fail when the independence assumption does not hold. In such situations it can be very useful to consider temporal/spatial statistical properties of the source signals as well. This has been done in form of suitable linear filtering [6] to achieve a sparse and independent representation of the signals. In [7] the author suggests to model the sources as a stochastic process and to do the ICA on the innovations rather than on the signals them self.

In this work we extend the ICA to multidimensional channels of neighboring realizations. The used data model is explained in detail in the following section. In section 3 it will be shown, that there are optimal features, that carry lower dependencies

between the sources and can be used for source separation. A heuristic is introduced, that allows to discard those features, that carry most of the dependencies. This leads to the Two-Step algorithm described in section 4. Our method requires (*i*) sources which exhibit correlations between neighboring pixels (e.g. continuous sources like images or sound signals), and (*ii*) sources from which sparse and almost independent features can be extracted. In section 5 we show separation results and benchmarks for linearly mixed passport photographs. The method is fast and provides good separation results even for sources, whose correlation coefficient is as large as 0.9.

## 2 Sources and observations

Let us consider a set of $N$ source signals $S_i(\mathbf{r})$, $i = 1, \ldots, N$ of length $L$, where $\mathbf{r}$ is a discrete sample index. The sample index could be of arbitrary dimension, but we assume that it belongs to some metric space so that neighborhood relations can be defined. The sample index might be a scalar for sources which are time series and a two-dimensional vector for sources which are images[1]. The sources are linearly combined by an unknown mixing matrix $\mathbf{A}$ of full rank to produce a set of $N$ observations $X_i(\mathbf{r})$,

$$ X_i(\mathbf{r}) = \sum_{j=1}^{N} A_{ij} S_j(\mathbf{r}) , \qquad (1) $$

and we assume that the mixing process is stationary, i.e. that the mixing matrix $\mathbf{A}$ is independent of $\mathbf{r}$. In the following we refer to the vectors $\mathbf{S}(\mathbf{r}) = (S_1(\mathbf{r}), \ldots, S_N(\mathbf{r}))^T$ and $\mathbf{X}(\mathbf{r}) = (X_1(\mathbf{r}), \ldots, X_N(\mathbf{r}))^T$ as a source and an observation stack. The goal is to find an appropriate demixing matrix $\mathbf{W}$ which - when applied to the observations $\mathbf{X}(\mathbf{r})$ - recovers good estimates $\hat{\mathbf{S}}(\mathbf{r})$,

$$ \hat{\mathbf{S}}(\mathbf{r}) = \mathbf{W}\mathbf{X}(\mathbf{r}) \approx \mathbf{S}(\mathbf{r}) \qquad (2) $$

of the original source signals (up to a permutation and scaling of the sources). Since the mixing matrix $\mathbf{A}$ is not known its inverse $\mathbf{W}$ has to be detected blindly, i.e. only properties of the sources which are detectable in the mixtures can be exploited. For a large class of ICA algorithms one assumes that the sources are non-Gaussian and independent, i.e. that the random vector $\mathbf{S}$ which is sampled by $L$ realizations

$$ \mathbf{S} : \{\mathbf{S}(\mathbf{r}_l), \, l = 1, \ldots, L\} \qquad (3) $$

has a factorizing and non-Gaussian joint probability distribution[2]. In situations, however, where the independence assumption does not hold, it can be helpful to take into account spatial dependencies, which can be very prominent for natural signals, and have been subject for a number of blind source separation algorithms [8, 9, 6]. Let us now consider patches $s_i(\mathbf{r})$,

$$ \mathbf{s}(\mathbf{r}) = \begin{pmatrix} s_1(\mathbf{r}) \\ \vdots \\ s_N(\mathbf{r}) \end{pmatrix} = \begin{pmatrix} S_1(\mathbf{r}+\delta\mathbf{r}_1) & \cdots & S_1(\mathbf{r}+\delta\mathbf{r}_M) \\ \vdots & \cdots & \vdots \\ S_N(\mathbf{r}+\delta\mathbf{r}_1) & \cdots & S_N(\mathbf{r}+\delta\mathbf{r}_M) \end{pmatrix} , \qquad (4) $$

of $M \ll L$ neighboring source samples. $s_i(\mathbf{r})$ could be a sequence of $M$ adjacent samples of an audio signal or a rectangular patch of $M$ pixels in an image. Instead of $L$ realizations of a random $N$-vector $\mathbf{S}$ (cf. eq. (3)) we now obtain a little less than $L$ realizations of a random $N \times M$ matrix $\mathbf{s}$,

$$\mathbf{s} : \{\mathbf{s}(\mathbf{r})\} . \tag{5}$$

Because of the stationarity of the mixing process we obtain

$$\mathbf{x} = \mathbf{A}\mathbf{s} \quad \text{and} \quad \hat{\mathbf{s}} = \mathbf{W}\mathbf{x}, \tag{6}$$

where $\mathbf{x}$ is an $N \times M$ matrix of neighboring observations and where the matrices $\mathbf{A}$ and $\mathbf{W}$ operate on every column vector of $\mathbf{s}$ and $\mathbf{x}$.

## 3  Optimal spatial features

Let us now consider a set of sources which are not statistically independent, i.e. for which

$$p(\mathbf{S}) = p(s_{1k}, \ldots, s_{Nk}) \neq \prod_{i=1}^{N} p(s_{ik}) \qquad \text{for all} \quad k = 1 \ldots M. \tag{7}$$

Our goal is to find in a first step a linear transformation $\mathbf{\Omega} \in \mathbb{R}^{M \times M}$ which - when applied to every patch - yields transformed sources $\mathbf{u} = \mathbf{s}\mathbf{\Omega}^T$ for which the independence assumption, $p(u_{1k}, \ldots, u_{Nk}) = \prod_{i=1}^{N} p(u_{ik})$ does hold for all $k = 1 \ldots M$, at least approximately. When $\mathbf{\Omega}$ is applied to the observations $\mathbf{x}$, $\mathbf{v} = \mathbf{x}\mathbf{\Omega}^T$, we obtain a modified source separation problem

$$\mathbf{u} = \mathbf{s}\mathbf{\Omega}^T = (\mathbf{W}\mathbf{x})\mathbf{\Omega}^T = \mathbf{W}(\mathbf{x}\mathbf{\Omega}^T) = \mathbf{W}\mathbf{v} , \tag{8}$$

where the demixing matrix $\mathbf{W}$ can be estimated from the transformed observations $\mathbf{v}$ in a second step using standard ICA. Eq. (7) is tantamount to positive trans-information of the source amplitudes.

$$I(s_{1k}, \ldots, s_{Nk}) = D_{KL}\left(p(s_{1k}, \ldots, s_{Nk}) \| \prod_{i=1}^{N} p(s_{ik})\right) > 0 , \tag{9}$$

where $D_{KL}$ is the Kullback-Leibler distance. As all elements of the patches are equally distributed, this quantity is the same for all $k$. Clearly, the dependencies, that are carried by single elements of the patches, are also present between whole patches, i.e. $I(s_1, s_2, \cdots, s_N) > 0$. However, since neighboring samples are correlated, it holds

$$I(s_1, s_2, \cdots, s_N) < \sum_{k=1}^{M} I(s_{1k}, s_{2k}, \cdots, s_{Nk}) . \tag{10}$$

Only if the sources where spatially white and $\mathbf{s}$ would consist of independent column vectors, this would hold with equality. When $\mathbf{\Omega}$ is applied to the source patches, the trans-information between patches is not changed, provided $\mathbf{\Omega}$ is a non-singular transformation. Neither information is introduced nor discarded by this transformation and it holds

$$I(s_1, s_2, \cdots, s_N) = I(s_1\mathbf{\Omega}^T, s_2\mathbf{\Omega}^T, \cdots, s_N\mathbf{\Omega}^T) . \tag{11}$$

For the optimal $\mathbf{\Omega}$ now the column vectors of $\mathbf{u} = \mathbf{s}\mathbf{\Omega}^T$ shall be independent. From (10) and (11) it follows that

$$I(u_1, u_2, \cdots, u_N) = \sum_{k=1}^{M} I(u_{1k}, u_{2k}, \cdots, u_{Nk}) < \sum_{k=1}^{M} I(s_{1k}, s_{2k}, \cdots, s_{Nk}) \ . \quad (12)$$

The column vectors of $\mathbf{u}$ are in general not equally distributed anymore, however the average trans-information has decreased to the level of information carried between the patches. In the experiments we shall see that this can be sufficiently small to reliably estimate the de-mixing matrix $\mathbf{W}$.

So it remains to estimate a matrix $\mathbf{\Omega}$ that provides a matrix $\mathbf{u}$ with independent columns. We approach this by estimating $\mathbf{\Omega}$ so that it provides row vectors of $\mathbf{u}$ that have independent elements, i.e. $p(u_i) = \prod_{k=1}^{M} p(u_{ik})$ for all $i$. With that and under the assumption that all sources may come from the same distribution and that there are no "cross dependencies" in $\mathbf{u}$ (i.e. $p(u_{ik})$ is independent from $p(u_{jl})$ for $k \neq l$), the independence is guaranteed also for whole column vectors of $\mathbf{u}$. Thus, standard ICA can be applied to patches of sources which yields $\mathbf{\Omega}$ as the de-mixing matrix. For real world applications however, $\mathbf{\Omega}$ has to be estimated from the observations $\mathbf{x}\mathbf{\Omega}^T = \mathbf{v}$. It holds the relation $\mathbf{v} = \mathbf{A}\mathbf{u}$, i.e. $\mathbf{A}$ only interchanges rows. So column vectors of $\mathbf{u}$ are independent to each other if, and only if columns of $\mathbf{v}$ are independent[3]. Thus, $\mathbf{\Omega}$ can be computed from $\mathbf{x}$ as well.

According to Eq. (12) the trans-information of the elements of columns of $\mathbf{u}$ has decreased in average, but not necessarily uniformly. One can expect some columns to have more independent elements than others. Thus, it may be advantageous to detect these columns rsp. the corresponding rows of $\mathbf{\Omega}$ and discard them prior to the second ICA step. Each source patch $s_i$ can be considered as linear combination of independent components, that are given by the columns of $\mathbf{\Omega}^{-1}$, where the elements of $u_i$ are the coefficients. In the result of the ICA, the coefficients have normalized variance. Therefore, those components, that have large Euklidian norm, occur as features with high entropy in the source patches. At the same time it is clear that, if there are features, that are responsible for the source dependencies, these features have to be present with large entropy, otherwise the source dependencies would have been low. Accordingly we propose a heuristic that discards the rows of $\mathbf{\Omega}$ with the smallest Euklidian norm prior to the second ICA step. How many rows have to be discarded and if this type of heuristic is applicable at all, depends of the statistical nature of the sources. In section 5 we show that for the test data this heuristic is well applicable and almost all dependencies are contained in one feature.

## 4  The Two-Step algorithm

The considerations of the previous section give rise to a Two-Step algorithm. In the first step the transformation $\mathbf{\Omega}$ has to be estimated. Standard ICA [1, 2, 5] is performed on $M$-dimensional patches, which are chosen with equal probability from all of the observed mixtures and at random positions. The positions may overlap but don't overlap the boundaries of the signals.

The resulting "demixing matrix" $\mathbf{\Omega}$ is applied to the patches of observations, generating a matrix $\mathbf{v}(\mathbf{r}) = \mathbf{x}(\mathbf{r})\mathbf{\Omega}^T$, the columns of which are candidates for the input for the second ICA. A number of $M^D$ columns that belong to rows of $\mathbf{\Omega}$ with small norm are discarded as they very likely represent features, that carry dependencies between the sources. $M^D$ is chosen as a model parameter or it can be determined empirically, given the data at hand (for instance by detecting a major jump in the

increase of the row norm of $\mathbf{\Omega}$). For the remaining columns it is not obvious which one represents the most sparse and independent feature. So any of them with equal probability now serve as input sample for the second ICA, which estimates the demixing matrix $\mathbf{W}$.

When the number $N$ of sources is large, the first ICA may fail to extract the independent source features, because, according to the central limit theorem, the distribution of their coefficients in the mixtures may be close to a Gaussian distribution. In such a situation we recommend to apply the abovementioned two steps repeatedly. The source estimates $\mathbf{Wx(r)}$ are used as input for the first ICA to achieve a better $\mathbf{\Omega}$, which in turn allows to better estimate $\mathbf{W}$.

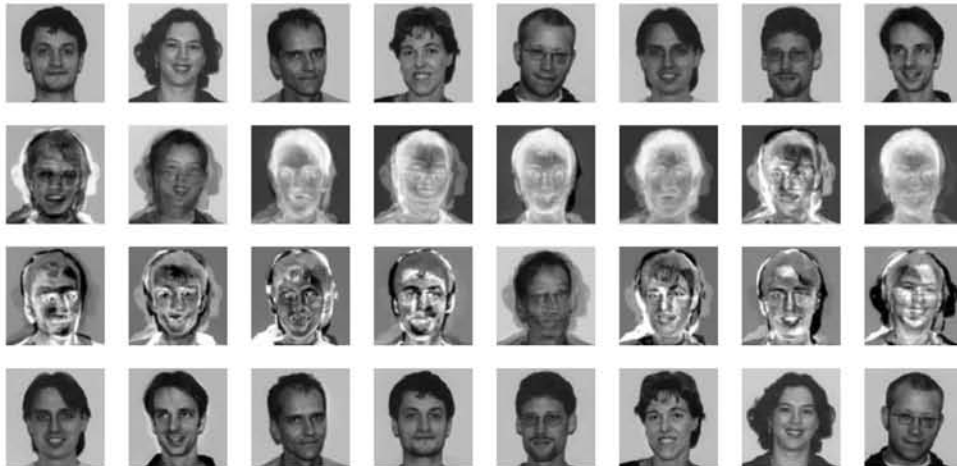

Figure 1: Results of standard and multidimensional ICA performed on a set of 8 correlated passport images. **Top row:** source images; **Second row:** linearly mixed sources; **Third row:** separation results using kurtosis optimization (*FastICA* Matlab package); **Bottom row:** separation results using multidimensional ICA (For explanation see text).

## 5   Numerical experiments

We applied our method to a linear mixture of 8 passport photographs which are shown in Fig. 1, top row. The images were mixed (cf. Fig. 1, second row) using a matrix whose elements were chosen randomly from a normal distribution with mean zero and variance one. The mixing matrix had a condition number of 80. The correlation coefficients of the source images were between 0.4 and 0.9 so that standard ICA methods failed to recover the sources: Fig. 1, 3rd row, shows the results of a kurtosis optimization using the *FastICA* Matlab package[4].

Fig. 1, bottom row, shows the result of the Two-Step multidimensional ICA described in section 4. For better comparison images were inverted manually to appear positive. In the first step $\mathbf{\Omega}$ was estimated using *FastICA* on $10^5$ patches, $6 \times 6$ pixels in size, which were taken with equal probability from random positions from all mixtures. The result of the first ICA is displayed in Fig. 2. The top row shows the row vectors of $\mathbf{\Omega}$ sorted by the logarithm of their norm. The second row shows the features (the corresponding columns of $\mathbf{\Omega}^{-1}$) which are extracted by $\mathbf{\Omega}$. In the dia-

gram below the stars indicate the logarithm of the row norm, $\log \sqrt{\sum_{l=1}^{M} \boldsymbol{\Omega}_{kl}^2}$, and the squares indicate the mutual information $I(u_{1k}, u_{7k})$ between the $k$-th features in sources 1 and 7 [5], calculated using a histogram estimator. It is quite prominent that $(i)$ a small norm of a column vector corresponds to a strongly correlated feature, and $(ii)$ there is only one feature which carries most of the dependencies between the sources. Thus, the first column of $\mathbf{v}$ was discarded. The second ICA was applied to any of the remaining components, chosen randomly and with equal probability. A comparison between Figs. 1, top and bottom rows, shows that all sources were successfully recovered.

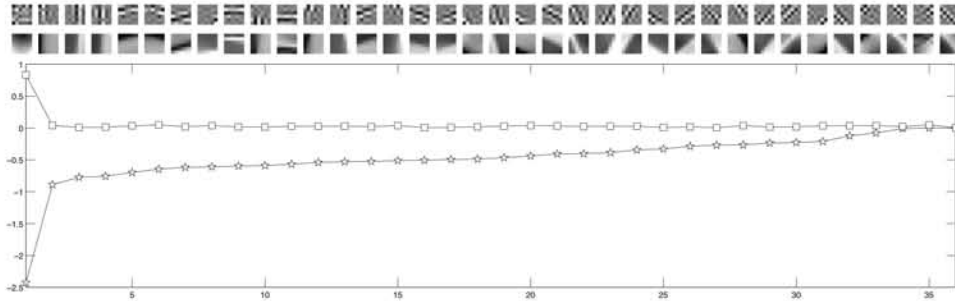

Figure 2: Result of an ICA (kurtosis optimization) performed on patches of observations (cf. Fig. 1, 2nd row), $6 \times 6$ pixels in size. **Top row:** Row vectors of the demixing matrix $\boldsymbol{\Omega}$. **Second row:** Corresponding column vectors of $\boldsymbol{\Omega}^{-1}$. Vectors are sorted by increasing norm of the row vectors; dark and bright pixels indicate positive and negative values. **Bottom diagram:** Logarithm of the norm of row vectors (stars) and mutual information $I(u_{1k}, u_{7k})$ (squares) between the coefficients of the corresponding features in the source images 1 and 7.

In the next experiment we examined the influence of selecting columns of $\mathbf{v}$ prior to the second ICA. In Fig. 3 we show the reconstruction error (cf. appendix A), that could be achieved with the second ICA when only a single column of $\mathbf{v}$ served as input. From the previous experiment we have seen, that only the first component has considerable dependencies. As expected, only the first column yields poor reconstruction error. Fig. 4 shows the reconstruction error vs. $M^D$ when the $M^D$ smallest norm rows of $\boldsymbol{\Omega}$ (rsp. columns of $\mathbf{v}$) are discarded. We see, that for all values a good reconstruction is achieved ($re < 0.6$). Even if no row is discarded the result is only slightly worse than for one or two discarded rows. The dependencies of the first component are "averaged" by the vast majority of components, that carry no dependencies, in this case. The conspicuous large variance of the error for larger numbers $M^D$ might be due to convergence instabilities or close to Gaussian distributed columns of $\mathbf{u}$. In either case it gives rise to discard as few components as possible. To evaluate the influence of the patch size $M$, the Two-Step algorithm was applied to 9 different mixtures of the sources shown in Fig. 1, top row, and using patch sizes between $M = 2 \times 2$ and $M = 6 \times 6$. Table 1 shows the mean and standard deviation of the achieved reconstruction error. The mixing matrix $\mathbf{A}$ was randomly chosen from a normal distribution with mean zero and variance one. *FastICA* was used for both steps, where $5 \cdot 10^5$ sample patches were used to extract the optimal features and $2.5 \cdot 10^4$ samples were used to estimate $\mathbf{W}$. The smallest row of $\boldsymbol{\Omega}$ was always discarded. The algorithm shows a quite robust performance, and even for patch sizes of $2 \times 2$ pixels a fairly good separation result is achieved

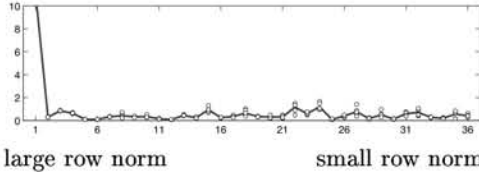

large row norm                     small row norm

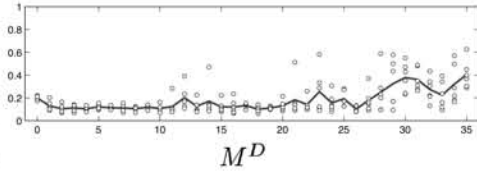

$M^D$

Figure 3: Every single row of $\mathbf{\Omega}$ used to generate input for the second ICA. Only the first (smallest norm) row causes bad reconstruction error for the second ICA step.

Figure 4: $M^D$ rows with smallest norm discarded. All values of $M^D$ provide good reconstruction error in the second step. Note the slidely worse result for $M^D = 0$ !

| patch size $M$ | $\mu_{re}$ | $\sigma_{re}$ |
|---|---|---|
| $2 \times 2$ | 0.4361 | 0.0383 |
| $3 \times 3$ | 0.2322 | 0.0433 |
| $4 \times 4$ | 0.1667 | 0.0263 |
| $5 \times 5$ | 0.1408 | 0.0270 |
| $6 \times 6$ | 0.1270 | 0.0460 |

Table 1: Separation result of the Two-Step algorithm performed on a set of 8 correlated passport images (cf. Fig. 1, top row). The table shows the average reconstruction error $\mu_{re}$ and its standard deviation $\sigma_{re}$ calculated from 9 different mixtures.

(Note, for comparison, that the reconstruction error of the separation in Fig. 1, bottom row, was 0.2).

## 6 Summary and outlook

We extended the source separation model to multidimensional channels (image patches). There are two linear transformations to be considered, one operating inside the channels ($\mathbf{\Omega}$) and one operating between the different channels ($\mathbf{W}$). The two transformations are estimated in two adjacent ICA steps. There are mainly two advantages, that can be taken from the first transformation: ($i$) By arranging independence among the columns of the transformed patches, the average transinformation between different channels is decreased. ($ii$) A suitable heuristic can be applied to discard those columns of the transformed patches, that carry most of the dependencies between different channels. A heuristic, that identifies the dependence carrying components by a small norm of the corresponding rows of $\mathbf{\Omega}$ has been introduced. It shows, that for the image data only one component carries most of the dependencies. Due this fact, the described method works well also when all components are taken into account. In future work, we are going to establish a Maximum Likelihood model for both transformations. We expect a performance gain due to the mutual improvement of the estimates of $\mathbf{W}$ and $\mathbf{\Omega}$ during the iterations. It remains to examine what the model has to be in case some rows of $\mathbf{\Omega}$ are discarded. In this case the transformations don't preserve the dimensionality of the observation patches.

## A Reconstruction error

The reconstruction error $re$ is a measure for the success of a source separation. It compares the estimated de-mixing matrix $\mathbf{W}$ with the inverse of the original mixing matrix $\mathbf{A}$ with respect to the indeterminacies: scalings and permutations. It is always nonnegative and equals zero if, and only if $\mathbf{P} = \mathbf{W}\mathbf{A}$ is a nonsingular

permutation matrix. This is the case when for every row of $\mathbf{P}$ exactly one element is different from zero and the rows of $\mathbf{P}$ are orthogonal, i.e. $\mathbf{PP}^T$ is a diagonal matrix. The reconstruction error is the sum of measures for both aspects

$$
\begin{aligned}
re &= 2\sum_{i=1}^{N}\log\sum_{j=1}^{N}\mathbf{P}_{ij}^{2} - \sum_{i=1}^{N}\log\sum_{j=1}^{N}\mathbf{P}_{ij}^{4} \; + \; \sum_{i=1}^{N}\log\sum_{j=1}^{N}\mathbf{P}_{ij}^{2} - \log\,\det\mathbf{PP}^{T} \\
&= 3\sum_{i=1}^{N}\log\sum_{j=1}^{N}\mathbf{P}_{ij}^{2} - \sum_{i=1}^{N}\log\sum_{j=1}^{N}\mathbf{P}_{ij}^{4} - \log\,\det\mathbf{PP}^{T}\,. \quad (13)
\end{aligned}
$$

**Acknowledgment:** This work was funded by the German Science Foundation (grant no. DFG SE 931/1-1 and DFG OB 102/3-1 ) and Wellcome Trust 061113/Z/00.

## Footnotes

[1]In the following we will mostly consider images, hence we will refer to the abovementioned neighborhood relations as *spatial relations*.

[2]In the following, symbols without sample index will refer to the random variable rather than to the particular realization.

[3]We assume non-Gaussian distributions for $\mathbf{u}$ and $\mathbf{v}$.

[4]http://www.cis.hut.fi/projects/ica/fastica/

[5] Images no. 1 and 7 were chosen exemplarily as the two most strongly correlated sources.

# References

[1] Anthony J. Bell and Terrence J. Sejnowski, "An information-maximization approach to blind separation and blind deconvolution," *Neural Computation*, vol. 7, no. 6, pp. 1129–1159, 1995.

[2] S. Amari, A. Cichocki, and H. H. Yang, "A new learning algorithm for blind signal separation," in *Advances in Neural Information Processing Systems*, D. S. Touretzky, M. C. Mozer, and M. E. Hasselmo, Eds., 1995, vol. 8.

[3] J. F. Cardoso, "Infomax and maximum likelihood for blind source separation," *IEEE Signal Processing Lett.*, 1997.

[4] Jean-François Cardoso, Sandip Bose, and Benjamin Friedlander, "On optimal source separation based on second and fourth order cumulants," in *Proc. IEEE Workshop on SSAP, Corfou, Greece*, 1996.

[5] A. Hyvärinen and E. Oja, "A fast fixed point algorithm for independent component analysis.," *Neural Comput.*, vol. 9, pp. 1483–1492, 1997.

[6] M. Zibulevski and B. A. Pearlmutter, "Blind source separation by sparse decomposition in a signal dictionary," *Neural Computation*, vol. 12, no. 3, pp. 863–882, April 2001.

[7] A. Hyvärinen, "Independent component analysis for time-dependent stochastic processes," in *Proc. Int. Conf. on Artificial Neural Networks (ICANN'98)*, 1998, pp. 541–546.

[8] L. Molgedey and H. G. Schuster, "Separation of a mixture of independent signals using time delayed correlations," *Phys. Rev. Lett.*, vol. 72, pp. 3634–3637, 1994.

[9] H. Attias and C. E. Schreiner, "Blind source separation and deconvolution: The dynamic component analysis algorithm," *Neural Comput.*, vol. 10, pp. 1373–1424, 1998.

[10] Anthony J. Bell and Terrence J. Sejnowski, "The 'independent components' of natural scenes are edge filters," *Vision Res.*, vol. 37, pp. 3327–3338, 1997.
